# Bayesian Image Super-resolution, Continued

**Lyndsey C. Pickup, David P. Capel**[†]**, Stephen J. Roberts Andrew Zisserman**
Information Engineering Building, Dept. of Eng. Science, Parks Road, Oxford, OX1 3PJ, UK
{elle,sjrob,az}@robots.ox.ac.uk
[†] 2D3, d.capel@2d3.com

## Abstract

This paper develops a multi-frame image super-resolution approach from a Bayesian view-point by marginalizing over the unknown registration parameters relating the set of input low-resolution views. In Tipping and Bishop's Bayesian image super-resolution approach [16], the marginalization was over the super-resolution image, necessitating the use of an unfavorable image prior. By integrating over the registration parameters rather than the high-resolution image, our method allows for more realistic prior distributions, and also reduces the dimension of the integral considerably, removing the main computational bottleneck of the other algorithm. In addition to the motion model used by Tipping and Bishop, illumination components are introduced into the generative model, allowing us to handle changes in lighting as well as motion. We show results on real and synthetic datasets to illustrate the efficacy of this approach.

## 1   Introduction

Multi-frame image super-resolution refers to the process by which a group of images of the same scene are fused to produce an image or images with a higher spatial resolution, or with more visible detail in the high spatial frequency features [7]. Such problems are common, with everything from holiday snaps and DVD frames to satellite terrain imagery providing collections of low-resolution images to be enhanced, for instance to produce a more aesthetic image for media publication [15], or for higher-level vision tasks such as object recognition or localization [5].

Limits on the resolution of the original imaging device can be improved by exploiting the relative sub-pixel motion between the scene and the imaging plane. No matter how accurate the registration estimate, there will be some residual uncertainty associated with the parameters [13]. We propose a scheme to deal with this uncertainty by integrating over the registration parameters, and demonstrate improved results on synthetic and real digital image data.

Image registration and super-resolution are often treated as distinct processes, to be considered sequentially [1, 3, 7]. Hardie *et al.* demonstrated that the low-resolution image registration can be updated using the super-resolution image estimate, and that this improves a *Maximum a Posteriori* (MAP) super-resolution image estimate [5]. More recently, Pickup *et al.* used a similar joint MAP approach to learn more general geometric and photometric registrations, the super-resolution image, and values for the prior's parameters simultaneously [12]. Tipping and Bishop's *Bayesian image super-resolution* work [16] uses a Maximum Likelihood (ML) point estimate of the registration parameters and the camera imaging blur, found by integrating the high-resolution image out of the registration problem and optimizing the marginal probability of the observed low-resolution images directly. This gives an improvement in the accuracy of the recovered registration (measured against known truth on synthetic data) compared to the MAP approach.

The image-integrating Bayesian super-resolution method [16] is extremely costly in terms of computation time, requiring operations that scale with the cube of the total number of high-resolution

pixels, severely limiting the size of the image patches over which they perform the registration (they use $9 \times 9$ pixel patches). The marginalization also requires a form of prior on the super-resolution image that renders the integral tractable, though priors such as Tipping and Bishop's chosen Gaussian form are known to be poor for tasks such as edge preservation, and much super-resolution work has employed other more favorable priors [2, 3, 4, 11, 14].

It is generally more desirable to integrate over the registration parameters rather than the super-resolution image, because it is the registration that constitutes the "nuisance parameters", and the super-resolution image that we wish to estimate. We derive a new view of Bayesian image super-resolution in which a MAP high-resolution image estimate is found by marginalizing over the uncertain registration parameters. Memory requirements are considerably lower than the image-integrating case; while the algorithm is more costly than a simple MAP super-resolution estimate, it is not infeasible to run on images of several hundred pixels in size.

Sections 2 and 3 develop the model and the proposed objective function. Section 4 evaluates results on synthetically-generated sequences (with ground truth for comparison), and on a real data example. A discussion of this approach and concluding remarks can be found in section 5.

## 2   Generative model

The generative model for multi-frame super-resolution assumes a known scene $\mathbf{x}$ (vectorized, size $N \times 1$), and a given registration vector $\boldsymbol{\theta}^{(k)}$. These are used to generate a vectorized low-resolution image $\mathbf{y}^{(k)}$ with $M$ pixels through a system matrix $\mathbf{W}^{(k)}$. Gaussian *i.i.d.* noise with precision $\beta$ is then added to $\mathbf{y}^{(k)}$,

$$\mathbf{y}^{(k)} = \lambda_\alpha^{(k)} \mathbf{W}\left(\boldsymbol{\theta}^{(k)}\right) \mathbf{x} + \boldsymbol{\lambda}_\beta^{(k)} + \boldsymbol{\epsilon}^{(k)} \tag{1}$$

$$\boldsymbol{\epsilon}^{(k)} \sim \mathcal{N}\left(\mathbf{0}, \beta^{-1}\mathbf{I}\right). \tag{2}$$

Photometric parameters $\lambda_\alpha$ and $\lambda_\beta$ provide a global affine correction for the scene illumination, and $\boldsymbol{\lambda}_\beta$ is simply an $M \times 1$ vector filled out with the value of $\lambda_\beta$. Each row of $\mathbf{W}^{(k)}$ constructs a single pixel in $\mathbf{y}^{(k)}$, and the row's entries are the vectorized and point-spread function (PSF) response for each low-resolution pixel, in the frame of the super-resolution image [2, 3, 16]. The PSF is usually assumed to be an isotropic Gaussian on the imaging plane, though for some motion models (*e.g.* planar projective) this does not necessarily lead to a Gaussian distribution on the frame of $\mathbf{x}$.

For an individual low-resolution image, given registrations and $\mathbf{x}$, the data likelihood is

$$p\left(\mathbf{y}^{(k)} \,\middle|\, \mathbf{x}, \boldsymbol{\theta}^{(k)}, \boldsymbol{\lambda}^{(k)}\right) = \left(\frac{\beta}{2\pi}\right)^{\frac{M}{2}} \exp\left\{-\frac{\beta}{2}\left\|\mathbf{y}^{(k)} - \lambda_\alpha^{(k)} \mathbf{W}\left(\boldsymbol{\theta}^{(k)}\right)\mathbf{x} - \boldsymbol{\lambda}_\beta^{(k)}\right\|_2^2\right\}. \tag{3}$$

When the registration is known approximately, for instance by pre-registering inputs, the uncertainty can be modeled as a Gaussian perturbation about the mean estimate $\bar{\boldsymbol{\theta}}^{(k)}$ for each image's parameter set, with covariance $\mathbf{C}$, which we restrict to be a diagonal matrix,

$$\begin{bmatrix} \boldsymbol{\theta}^{(k)} \\ \lambda_\alpha^{(k)} \\ \lambda_\beta^{(k)} \end{bmatrix} = \begin{bmatrix} \bar{\boldsymbol{\theta}}^{(k)} \\ \bar{\lambda}_\alpha^{(k)} \\ \bar{\lambda}_\beta^{(k)} \end{bmatrix} + \boldsymbol{\delta}^{(k)} \tag{4}$$

$$\boldsymbol{\delta}^{(k)} \sim \mathcal{N}\left(\mathbf{0}, \mathbf{C}\right) \tag{5}$$

$$p\left(\boldsymbol{\theta}^{(k)}, \boldsymbol{\lambda}^{(k)}\right) = \left(\frac{|\mathbf{C}^{-1}|}{(2\pi)^n}\right)^{\frac{1}{2}} \exp\left\{-\frac{1}{2}\boldsymbol{\delta}^{(k)T}\mathbf{C}^{-1}\boldsymbol{\delta}^{(k)}\right\}. \tag{6}$$

A Huber prior is assumed for the directional image gradients $\mathbf{Dx}$ in the super-resolution image $\mathbf{x}$ (in the horizontal, vertical, and two diagonal directions),

$$p\left(\mathbf{x}\right) = \frac{1}{\mathcal{Z}_x} \exp\left\{-\frac{\nu}{2}\rho\left(\mathbf{Dx}, \alpha\right)\right\} \tag{7}$$

$$\rho(z, \alpha) = \begin{cases} z^2 & \text{if } |z| < \alpha \\ 2\alpha|z| - \alpha^2 & \text{otherwise} \end{cases} \tag{8}$$

where $\alpha$ is a parameter of the Huber potential function, and $\nu$ is the prior strength parameter. This belongs to a family of functions often favored over Gaussians for super-resolution image priors [2, 3, 14] because the Huber distribution's heavy tails mean image edges are penalized less severely. The difficulty in computing the partition function $\mathcal{Z}_x$ is a consideration when marginalizing over $\mathbf{x}$ as in [16], though for the MAP image estimate, a value for this scale factor is not required.

Regardless of the exact forms of these probability distributions, probabilistic super-resolution algorithms can usually be interpreted in one of the following ways.

The most popular approach to super-resolution is to obtain a MAP estimate, typically using an iterative scheme to maximize $p\left(\mathbf{x}\left|\left\{\mathbf{y}^{(}k),\boldsymbol{\theta}^{(k)},\boldsymbol{\lambda}^{(k)}\right\}\right.\right)$ with respect to $\mathbf{x}$, where

$$p\left(\mathbf{x}\left|\left\{\mathbf{y}^{(}k),\boldsymbol{\theta}^{(k)},\boldsymbol{\lambda}^{(k)}\right\}\right.\right) = \frac{p\left(\mathbf{x}\right)\prod_{k=1}^{K}p\left(\mathbf{y}^{(k)}\left|\mathbf{x},\boldsymbol{\theta}^{(k)},\boldsymbol{\lambda}^{(k)}\right.\right)}{p\left(\left\{\mathbf{y}^{(k)}\right\}\left|\left\{\boldsymbol{\theta}^{(k)},\boldsymbol{\lambda}^{(k)}\right\}\right.\right)}, \tag{9}$$

and the denominator is an unknown scaling factor.

Tipping and Bishop's approach takes an ML estimate of the registration by marginalizing over $\mathbf{x}$, then calculates the super-resolution estimate as in (9). While Tipping and Bishop did not include a photometric model, the equivalent expression to be maximized with respect to $\boldsymbol{\theta}$ and $\boldsymbol{\lambda}$ is

$$p\left(\left\{\mathbf{y}^{(y)}\right\}\left|\left\{\boldsymbol{\theta}^{(k)},\boldsymbol{\lambda}^{(k)}\right\}\right.\right) = \int p\left(\mathbf{x}\right)\prod_{k=1}^{K}p\left(\mathbf{y}^{(y)}\left|\mathbf{x},\boldsymbol{\theta}^{(k)},\boldsymbol{\lambda}^{(k)}\right.\right)d\mathbf{x}. \tag{10}$$

Note that Tipping and Bishop's work does employ the same data likelihood expression as in (3), which forced them to select a Gaussian form for $p\left(\mathbf{x}\right)$, rather than a more suitable image prior, in order to keep the integral tractable.

Finally, in this paper we find $\mathbf{x}$ through marginalizing over $\boldsymbol{\theta}$ and $\boldsymbol{\lambda}$, so that a MAP estimate of $\mathbf{x}$ can be obtained by maximizing $p\left(\mathbf{x}\left|\left\{\mathbf{y}^{(k)}\right\}\right.\right)$ directly with respect to $\mathbf{x}$. This is achieved by finding

$$p\left(\mathbf{x}\left|\left\{\mathbf{y}^{(k)}\right\}\right.\right) = \frac{p(\mathbf{x})}{p\left(\left\{\mathbf{y}^{(k)}\right\}\right)}\int\prod_{k=1}^{K}p\left(\boldsymbol{\theta}^{(k)},\boldsymbol{\lambda}^{(k)}\right)p\left(\mathbf{y}^{(k)}\left|\mathbf{x},\boldsymbol{\theta}^{(k)},\boldsymbol{\lambda}^{(k)}\right.\right)d\left\{\boldsymbol{\theta},\boldsymbol{\lambda}\right\}, \tag{11}$$

which is developed further in the next section. Note that the integral does not involve the prior, $p\left(\mathbf{x}\right)$.

## 3   Marginalizing over registration parameters

In order to obtain an expression for $p\left(\mathbf{x}\left|\left\{\mathbf{y}^{(k)}\right\}\right.\right)$ from expressions (3), (6) and (7) above, the parameter variations $\boldsymbol{\delta}^{(k)}$ must be integrated out of the problem. Registration estimates $\bar{\boldsymbol{\theta}}^{(k)}$, $\bar{\lambda}_{\alpha}$ and $\bar{\lambda}_{\beta}$ can be obtained using classical registration methods, either intensity-based [8] or estimation from image points [6], and the diagonal matrix $\mathbf{C}$ is constructed to reflect the confidence in each parameter estimate. This might mean a standard deviation of a tenth of a low-resolution pixel on image translation parameters, or a few gray levels' shift on the illumination model, for instance.

The integral performed is

$$p\left(\mathbf{x}\left|\left\{\mathbf{y}^{(k)}\right\}\right.\right) = \frac{1}{p\left(\left\{\mathbf{y}^{(k)}\right\}\right)}\left(\frac{\beta}{2\pi}\right)^{\frac{KM}{2}}\left(\frac{b}{2\pi}\right)^{\frac{Kn}{2}}\frac{1}{\mathcal{Z}_x}\exp\left\{-\frac{\nu}{2}\rho\left(\mathbf{D}\mathbf{x},\alpha\right)\right\}$$

$$\times\int\exp\left\{-\sum_{k=1}^{K}\left[\frac{\beta}{2}\left\|\mathbf{y}^{(k)}-\lambda_{\alpha}^{(k)}\mathbf{W}\left(\boldsymbol{\theta}^{(k)}\right)\mathbf{x}-\boldsymbol{\lambda}_{\beta}^{(k)}\right\|_{2}^{2}+\frac{1}{2}\boldsymbol{\delta}^{(k)}\mathbf{C}^{(k)-1}\boldsymbol{\delta}^{(k)}\right]\right\}d\boldsymbol{\delta}, \tag{12}$$

where $\boldsymbol{\delta}^{T} = \left[\boldsymbol{\delta}^{(1)T},\boldsymbol{\delta}^{(2)T},\ldots,\boldsymbol{\delta}^{(K)T}\right]$ and all the $\boldsymbol{\lambda}$ and $\boldsymbol{\theta}$ parameters are functions of $\boldsymbol{\delta}$ as in (4). Expanding the data error term in the exponent for each low-resolution image as a second-order Taylor series about the estimated geometric registration parameter yields

$$e^{(k)}\left(\boldsymbol{\delta}\right) = \left\|\mathbf{y}^{(k)}-\lambda_{\alpha}\left(\boldsymbol{\theta}^{(k)}\right)\mathbf{W}^{(k)}\left(\boldsymbol{\theta}^{(k)}\right)\mathbf{x}-\lambda_{\beta}^{(k)}\left(\boldsymbol{\theta}^{(k)}\right)\right\|_{2}^{2} \tag{13}$$

$$= F^{(k)}+\mathbf{G}^{(k)T}\boldsymbol{\delta}+\frac{1}{2}\boldsymbol{\delta}^{(k)T}\mathbf{H}^{(k)}\boldsymbol{\delta}^{(k)}, \tag{14}$$

Values for $F$, $\mathbf{G}$ and $\mathbf{H}$ can be found numerically (for geometric registrations) or analytically (for the photometric parameters) from $\mathbf{x}$ and $\left\{\mathbf{y}^{(k)}, \boldsymbol{\theta}^{(k)}, \lambda_\alpha^{(k)}, \lambda_\beta^{(k)}\right\}$. Thus the whole exponent of (12), $f$, becomes

$$
f \;=\; \sum_{k=1}^{K} \left( -\frac{\beta}{2} F^{(k)} - \frac{\beta}{2} \mathbf{G}^{(k)T} \boldsymbol{\delta}^{(k)} - \frac{1}{2} \boldsymbol{\delta}^{(k)T} \left[ \frac{\beta}{2} \mathbf{H}^{(k)} + \mathbf{C}^{-1} \right] \boldsymbol{\delta}^{(k)} \right) \tag{15}
$$

$$
=\; -\frac{\beta}{2} F - \frac{\beta}{2} \mathbf{G}^T \boldsymbol{\delta} - \frac{1}{2} \boldsymbol{\delta}^T \left[ \frac{\beta}{2} \mathbf{H} + \mathbf{V}^{-1} \right] \boldsymbol{\delta}, \tag{16}
$$

where the omission of image superscripts indicates stacked matrices, and $\mathbf{H}$ is therefore a block-diagonal $nK \times nK$ sparse matrix, and $\mathbf{V}$ is comprised of the repeated diagonal of $\mathbf{C}$.

Finally, letting $\mathbf{S} = \frac{\beta}{2}\mathbf{H} + \mathbf{V}^{-1}$,

$$
\int \exp\{f\} \, d\boldsymbol{\delta} \;=\; \exp\left\{ -\frac{\beta}{2} F \right\} \int \exp\left\{ -\frac{\beta}{2} \mathbf{G}^T \boldsymbol{\delta} - \frac{1}{2} \boldsymbol{\delta}^T \mathbf{S} \boldsymbol{\delta} \right\} d\boldsymbol{\delta} \tag{17}
$$

$$
=\; \exp\left\{ -\frac{\beta}{2} F \right\} (2\pi)^{\frac{nK}{2}} |\mathbf{S}|^{-\frac{1}{2}} \exp\left\{ \frac{\beta^2}{8} \mathbf{G}^T \mathbf{S}^{-1} \mathbf{G} \right\}. \tag{18}
$$

The objective function, $\mathcal{L}$, to be minimized with respect to $\mathbf{x}$ is obtained by taking the negative log of (12), using the result from (18), and neglecting the constant terms:

$$
\mathcal{L} \;=\; \frac{\nu}{2} \rho\left(\mathbf{Dx}, \alpha\right) + \frac{\beta}{2} F + \frac{1}{2} \log |\mathbf{S}| - \frac{\beta^2}{8} \mathbf{G}^T \mathbf{S}^{-1} \mathbf{G}. \tag{19}
$$

This can be optimized using *Scaled Conjugate Gradients* (SCG) [9], noting that the gradient can be expressed

$$
\frac{d\mathcal{L}}{d\mathbf{x}} \;=\; \frac{\nu}{2} \mathbf{D}^T \frac{d}{d\mathbf{x}} \rho\left(\mathbf{Dx}\right) + \frac{\beta}{2} \frac{dF}{d\mathbf{x}} - \frac{\beta^2}{4} \mathbf{G}^T \mathbf{S}^{-1} \frac{d\mathbf{G}}{d\mathbf{x}}
$$
$$
+ \left[ \frac{\beta}{4} \text{vec}\left(\mathbf{S}^{-1}\right)^T - \frac{\beta^3}{16} \left(\mathbf{G}^T \mathbf{S}^{-1} \otimes \mathbf{G}^T \mathbf{S}^{-1}\right) \right] \frac{d \text{vec} \mathbf{H}}{d\mathbf{x}}, \tag{20}
$$

where derivatives of $F$, $\mathbf{G}$ and $\mathbf{H}$ with respect to $\mathbf{x}$ can be found analytically for photometric parameters, and numerically (using the analytic gradient of $e^{(k)}\left(\boldsymbol{\delta}^{(k)}\right)$ with respect to $\mathbf{x}$) with respect to the geometric parameters.

## 3.1 Implementation notes

Notice that the value $F$ from (16) is simply the reprojection error of the current estimate of $\mathbf{x}$ at the mean registration parameter values, and that gradients of this expression with respect to the $\boldsymbol{\lambda}$ parameters, and with respect to $\mathbf{x}$ can both be found analytically. To find the gradient with respect to a geometric registration parameter $\theta_i^{(k)}$, and elements of the Hessian involving it, a central difference scheme involving only the $k^{\text{th}}$ image is used.

Mean values for the registration are computed by standard registration techniques, and $\mathbf{x}$ is initialized using around 10 iterations of SCG to find the maximum likelihood solution evaluated at these mean parameters. Additionally, pixel values are scaled to lie between $-\frac{1}{2}$ and $\frac{1}{2}$, and the ML solution is bounded to lie within these values in order to curb the severe overfitting usually observed in ML super-resolution results.

In our implementation, the parameters representing the $\lambda$ values are scaled so that they share the same standard deviations as the $\boldsymbol{\theta}$ parameters, which represent the sub-pixel geometric registration shifts, which makes the matrix $\mathbf{V}$ a multiple of the identity. The scale factors are chosen so that one standard deviation in $\lambda_\beta$ gives a 10-gray-level shift, and one standard deviation in $\lambda_\alpha$ varies pixel values by around 10 gray levels at mean image intensity.

# 4 Results

The first experiment takes a sixteen-image synthetic dataset created from an eyechart image. Data is generated at a zoom factor of 4, using a 2D translation-only motion model, and the two-parameter global affine illumination model described above, giving a total of four registration parameters per low-resolution image. Gaussian noise with standard deviation equivalent to 5 gray levels is added to each low-resolution pixel independently. The sub-pixel perturbations are evenly spaced over a grid up to plus or minus one half of a low-resolution pixel, giving a similar setup to that described in [10], but with additional lighting variation. The ground truth image and two of the low-resolution images appear in the first row of Figure 1.

Geometric and photometric registration parameters were initialized to the identity, and the images were registered using an iterative intensity-based scheme. The resulting parameter values were used to recover two sets of super-resolution images: one using the standard Huber MAP algorithm, and the second using our extension integrating over the registration uncertainty. The Huber parameter $\alpha$ was fixed at $0.01$ for all runs, and $\nu$ was varied over a range of possible values representing ratios between $\nu$ and the image noise precision $\beta$.

The images giving lowest RMS error from each set are displayed in the second row of Figure 1. Visually, the differences between the images are subtle, though the bottom row of letters is better defined in the output from the new algorithm. Plotting the RMSE as a function of $\nu$ in Figure 2, we see that the proposed registration-integrating approach achieves a lower error, compared to the ground truth high-resolution image, than the standard Huber MAP algorithm for any choice of prior strength, $\nu$ in the optimal region.

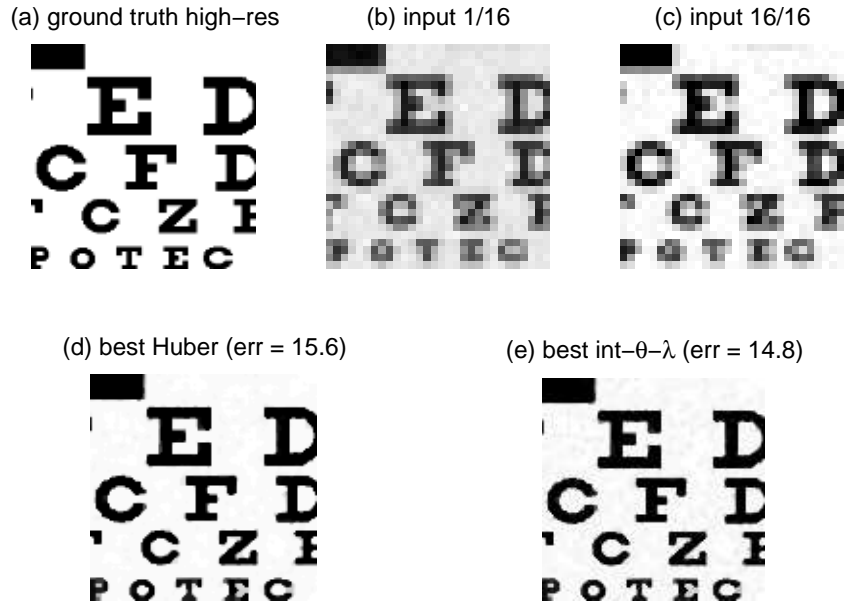

Figure 1: (a) Ground truth image. Only the central recoverable part is shown; (b,c) low-resolution images. The variation in intensity is clearly visible, and the sub-pixel displacements necessary for multi-frame image super-resolution are most apparent on the "D" characters to the right of each image; (d) The best (i.e. minimum MSE – see Figure 2) image from the regular Huber MAP algorithm, having super-resolved the dataset multiple times with different prior strength settings; (e) The best result using out approach of integrating over $\theta$ and $\lambda$. As well as having a lower RMSE, note the improvement in black-white edge detail on some of the letters on the bottom line.

The second experiment uses real data with a 2D translation motion model and an affine lighting model exactly as above. The first and last images appear on the top row of Figure 3. Image registration was carried out in the same manner as before, and the geometric parameters agree with the provided homographies to within a few hundredths of a pixel. Super-resolution images were created

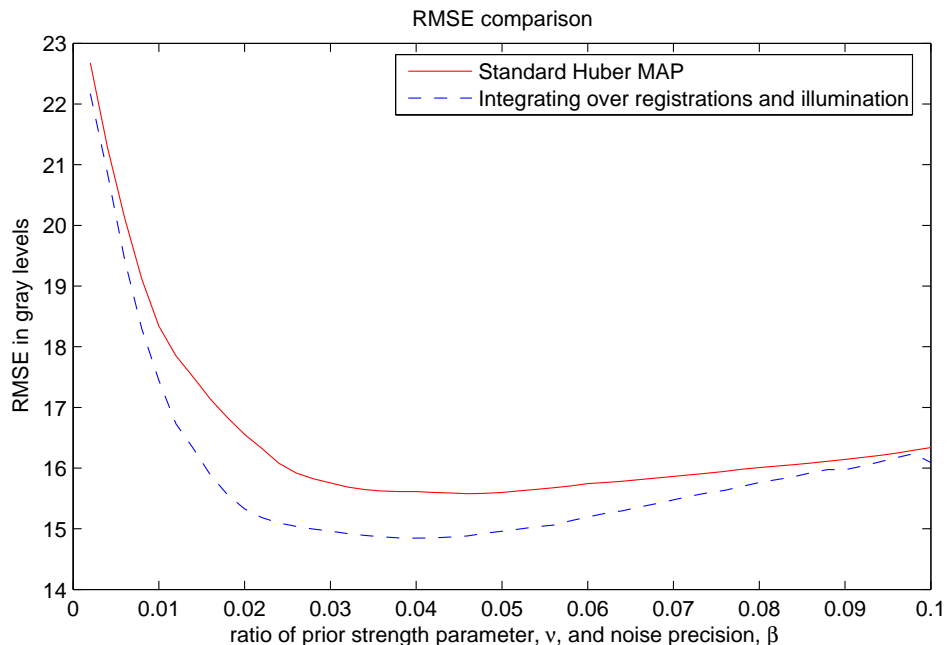

Figure 2: Plot showing the variation of RMSE with prior strength for the standard Huber-prior MAP super-resolution method and our approach integrating over $\theta$ and $\lambda$. The images corresponding to the minima of the two curves are shown in Figure 1

for a number of $\nu$ values, the equivalent values to those quoted in [3] were found subjectively to be the most suitable.

The covariance of the registration values was chosen to be similar to that used in the synthetic experiments. Finally, Tipping and Bishop's method was extended to cover the illumination model and used to register and super-resolve the dataset, using the same PSF standard deviation (0.4 low-resolution pixels) as the other methods.

The three sets of results on the real data sequence are shown in the middle and bottom rows of Figure 3. To facilitate a better comparison, a sub-region of each is expanded to make the letter details clearer. The Huber prior tends to make the edges unnaturally sharp, though it is very successful at regularizing the solution elsewhere. Between the Tipping and Bishop image and the registration-integrating approach, the text appears more clear in our method, and the regularization in the constant background regions is slightly more successful.

## 5 Discussion

It is possible to interpret the extra terms introduced into the objective function in the derivation of this method as an extra regularizer term or image prior. Considering (19), the first two terms are identical to the standard MAP super-resolution problem using a Huber image prior. The two additional terms constitute an additional distribution over $\mathbf{x}$ in the cases where $\mathbf{S}$ is not dominated by $\mathbf{V}$; as the distribution over $\theta$ and $\lambda$ tightens to a single point, the terms tend to constant values.

The intuition behind the method's success is that this extra prior resulting from the final two terms of (19) will favor image solutions which are not acutely sensitive to minor adjustments in the image registration. The images of figure 4 illustrate the type of solution which would score poorly. To create the figure, one dataset was used to produce two super-resolved images, using two independent sets of registration parameters which were randomly perturbed by an *i.i.d.* Gaussian vector with a standard deviation of only 0.04 low-resolution pixels. The checker-board pattern typical of ML super-resolution images can be observed, and the difference image on the right shows the drastic contrast between the two image estimates.

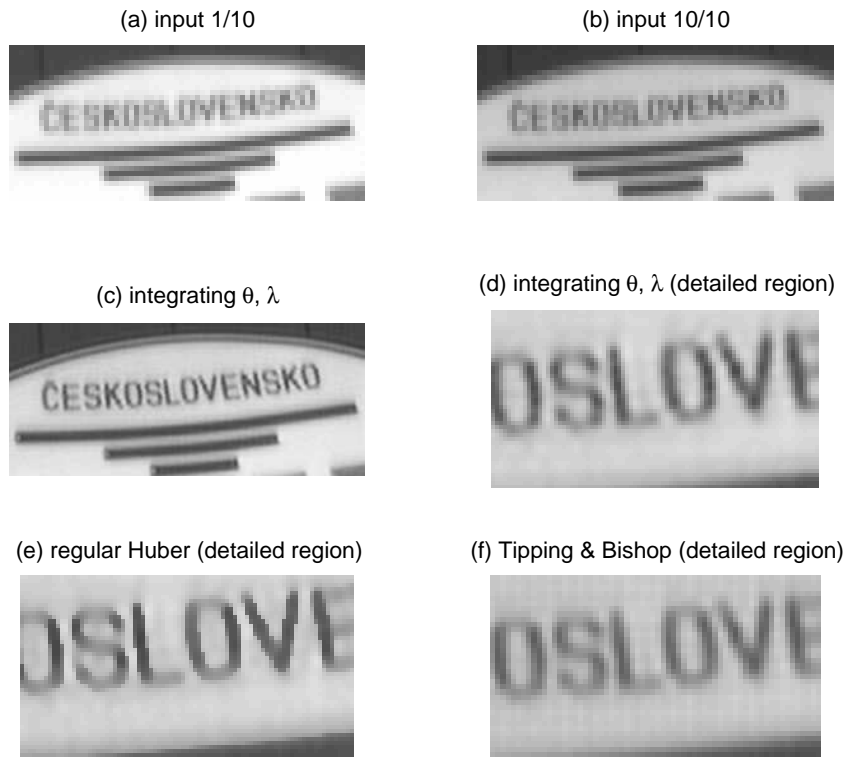

(a) input 1/10          (b) input 10/10

(c) integrating θ, λ          (d) integrating θ, λ (detailed region)

(e) regular Huber (detailed region)          (f) Tipping & Bishop (detailed region)

Figure 3: (a,b) First and last images from a real data sequence containing 10 images acquired on a rig which constrained the motion to be pure translation in 2D. (c) The full super-resolution output from our algorithm. (d) Detailed region of the central letters, again with our algorithm. (e) Detailed region of the regular Huber MAP super-resolution image, using parameter values suggested in [3], which are also found to be subjectively good choices. The edges are slightly artificially crisp, but the large smooth regions are well regularized. (f) Close-up of letter detail for comparison with Tipping and Bishop's method of marginalization. The Gaussian form of their prior leads to a more blurred output, or one that over-fits to the image noise on the input data if the prior's influence is decreased.

## 5.1   Conclusion

This work has developed an alternative approach for Bayesian image super-resolution with several advantages over Tipping and Bishop's original algorithm. These are namely a formal treatment of registration uncertainty, the use of a much more realistic image prior, and the computational speed and memory efficiency relating to the smaller dimension of the space over which we integrate. The results on real and synthetic images with this method show an advantage over the popular MAP approach, and over the result from Tipping and Bishop's method, largely owing to our more favorable prior over the super-resolution image.

It will be a straightforward extension of the current approach to incorporate learning for the point-spread function covariance, though it will result in a less sparse Hessian matrix $\mathbf{H}$, because each row and column associated with the PSF parameter(s) has the potential to be full-rank, assuming a common camera configuration is shared across all the frames.

Finally, the best way of learning the appropriate covariance values for the distribution over $\boldsymbol{\theta}$ given the observed data, and how to assess the trade-off between its "prior-like" effects and the need for a standard Huber-style image prior, are still open questions.

## Acknowledgements

The real dataset used in the results section is due to Tomas Pajdla and Daniel Martinec, CMP, Prague, and is available at `http://www.robots.ox.ac.uk/~vgg/data4.html`.

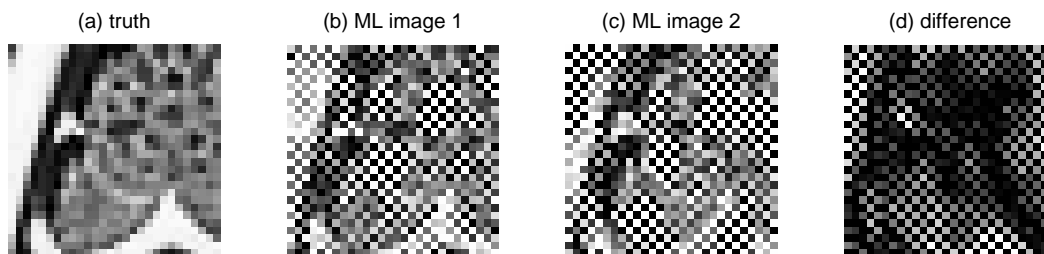

| (a) truth | (b) ML image 1 | (c) ML image 2 | (d) difference |

Figure 4: An example of the effect of tiny changes in the registration parameters. (a) Ground truth image from which a 16-image low-resolution dataset was generated. (b,c) Two ML super-resolution estimates. In both cases, the same dataset was used, but the registration parameters were perturbed by an *i.i.d.* vector with standard deviation of just 0.04 low-resolution pixels. (d) The difference between the two solutions. In all these images, values outside the valid image intensity range have been rounded to white or black values.

This work was funded in part by EC Network of Excellence PASCAL.

# References

[1] S. Baker and T. Kanade. Limits on super-resolution and how to break them. *IEEE Transactions on Pattern Analysis and Machine Intelligence*, 24(9):1167–1183, 2002.

[2] S. Borman. *Topics in Multiframe Superresolution Restoration*. PhD thesis, University of Notre Dame, Notre Dame, Indiana, May 2004.

[3] D. Capel. *Image Mosaicing and Super-resolution (Distinguished Dissertations)*. Springer, ISBN: 1852337710, 2004.

[4] S. Farsiu, M. Elad, and P. Milanfar. A practical approach to super-resolution. In *Proc. of the SPIE: Visual Communications and Image Processing*, San-Jose, 2006.

[5] R. C. Hardie, K. J. Barnard, and E. A. Armstrong. Joint map registration and high-resolution image estimation using a sequence of undersampled images. *IEEE Transactions on Image Processing*, 6(12):1621–1633, 1997.

[6] R. I. Hartley and A. Zisserman. *Multiple View Geometry in Computer Vision*. Cambridge University Press, ISBN: 0521540518, second edition, 2004.

[7] M. Irani and S. Peleg. Super resolution from image sequences. *ICPR*, 2:115–120, June 1990.

[8] M. Irani and S. Peleg. Improving resolution by image registration. *Graphical Models and Image Processing*, 53:231–239, 1991.

[9] I. Nabney. *Netlab algorithms for pattern recognition*. Springer, 2002.

[10] N. Nguyen, P. Milanfar, and G. Golub. Efficient generalized cross-validation with applications to parametric image restoration and resolution enhancement. *IEEE Transactions on Image Processing*, 10(9):1299–1308, September 2001.

[11] L. C. Pickup, S. J. Roberts, and A. Zisserman. A sampled texture prior for image super-resolution. In *Advances in Neural Information Processing Systems*, pages 1587–1594, 2003.

[12] L. C. Pickup, S. J. Roberts, and A. Zisserman. Optimizing and learning for super-resolution. In *Proceedings of the British Machine Vision Conference*, 2006. to appear.

[13] D. Robinson and P. Milanfar. Fundamental performance limits in image registration. *IEEE Transactions on Image Processing*, 13(9):1185—1199, September 2004.

[14] R. R. Schultz and R. L. Stevenson. A bayesian approach to image expansion for improved definition. *IEEE Transactions on Image Processing*, 3(3):233–242, 1994.

[15] Salient Stills. http://www.salientstills.com/.

[16] M. E. Tipping and C. M. Bishop. Bayesian imge super-resolution. In S. Thrun, S. Becker, and K. Obermayer, editors, *Advances in Neural Information Processing Systems*, volume 15, pages 1279–1286, Cambridge, MA, 2003. MIT Press.
